# Development of Orientation and Ocular Dominance Columns in Infant Macaques

**Klaus Obermayer**
Howard Hughes Medical Institute
Salk-Institute
La Jolla, CA 92037

**Lynne Kiorpes**
Center for Neural Science
New York University
New York, NY 10003

**Gary G. Blasdel**
Department of Neurobiology
Harvard Medical School
Boston, MA 02115

## Abstract

Maps of orientation preference and ocular dominance were recorded optically from the cortices of 5 infant macaque monkeys, ranging in age from 3.5 to 14 weeks. In agreement with previous observations, we found that basic features of orientation and ocular dominance maps, as well as correlations between them, are present and robust by 3.5 weeks of age. We did observe changes in the strength of ocular dominance signals, as well as in the spacing of ocular dominance bands, both of which increased steadily between 3.5 and 14 weeks of age. The latter finding suggests that the adult spacing of ocular dominance bands depends on cortical growth in neonatal animals. Since we found no corresponding increase in the spacing of orientation preferences, however, there is a possibility that the orientation preferences of some cells change as the cortical surface expands. Since correlations between the patterns of orientation selectivity and ocular dominance are present at an age, when the visual system is still immature, it seems more likely that their development may be an innate process and may not require extensive visual experience.

# 1    INTRODUCTION

Over the past years, high-resolution images of the simultaneous representation of orientation selectivity and ocular dominance have been obtained in large areas of macaque striate cortex using optical techniques [3, 4, 5, 6, 12, 18]. These studies confirmed that ocular dominance and orientation preference are organized in large parts in slabs. While optical recordings of ocular dominance are in accordance with previous findings, it turned out that iso-orientation slabs are much shorter than expected, and that the orientation map contains several other important elements of organization – singularities, fractures, and saddle-points.

A comparison between maps of orientation preference and ocular dominance, which were derived from the same region of adult monkey striate cortex, showed a pronounced relationship between both patterns [5, 12, 13, 15, 17]. Fourier analyses, for example, reveal that orientation preferences repeat at closer intervals along the ocular dominance slabs than they do across them. Singularities were found to align with the centers of ocular dominance bands, and the iso-orientation bands, which connect them, intersect the borders of ocular dominance bands preferably at angles close to $90°$.

Given the fact that these relationships between the maps of orientation and ocular dominance are present in all maps recorded from adult macaques, one naturally wonders how this organization matures. If the ocular dominance slabs were to emerge initially, for example, the narrower slabs of iso-orientation might later develop in between. This might seem likely given the anatomical segregation which is apparent for ocular dominance but not for orientation [9]. However, this possibility is contradicted by physiological studies that show normal, adult-like sequences of orientation preference in the early postnatal weeks in macaque when ocular dominance slabs are still immature [19]. The latter findings suggest a different developmental hypothesis; that the organization into regions of different orientation preferences may precede or even guide ocular dominance formation. A third possibility, consistent with both previous results, is that orientation and ocular dominance maps form independently and align in later stages of development.

In order to provide evidence for one or the other hypothesis, we investigated the relationship between ocular dominance and orientation preference in very young macaque monkeys. Results are presented in the remainder of this paper. Section 2 contains an overview about the experimental data, and section 3 relates the data to previous modelling efforts.

# 2    ORIENTATION AND OCULAR DOMINANCE COLUMNS IN INFANT MACAQUES

## 2.1    THE OVERALL STRUCTURE

Figure 1 shows the map of orientation preference (Fig. 1a) and ocular dominance (Fig. 1b) recorded from area 17 of a 3.5 week old macaque.[1] Both maps look similar

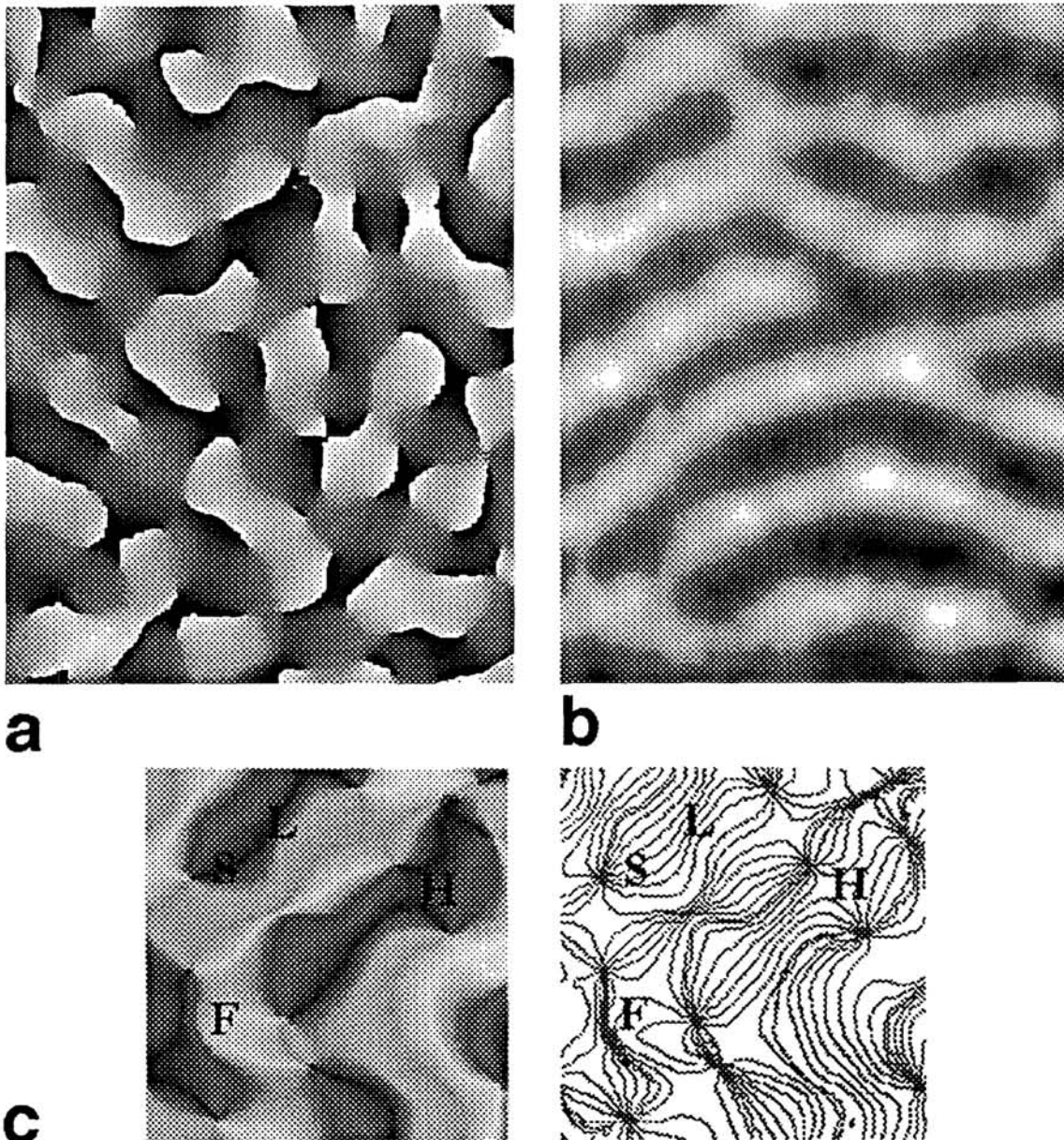

Figure 1: Spatial pattern of orientation preference and ocular dominance recorded from area 17 of a macaque, 3.5 weeks of age. Figures (a) and (b) show orientation preferences and ocular dominance bands within the same 3.1 mm × 4.3 mm large region of striate cortex. Brightness values in Fig. (a) indicate orientation preferences, where the interval of 180° is represented by the progression in colors from black to white. Brightness values in Fig. (b) indicate ocular dominance, where bright and dark denote ipsi- and contralateral eye-preference. respectively. The data was recorded from a region close to the border to area 18 and close to midline. Figure (c) shows an enlarged section of this map in the preference (left) and the in contour plot (right) representations. Iso-orientation lines on the right indicate intervals of 11.25°. Letters indicate linear zones (L), saddle points (H), singularities (S), and fractures (F).

to maps which have been recorded from adults. The orientation map exhibits all of the local elements which have been described [12, 13]: linear zones, saddle points,

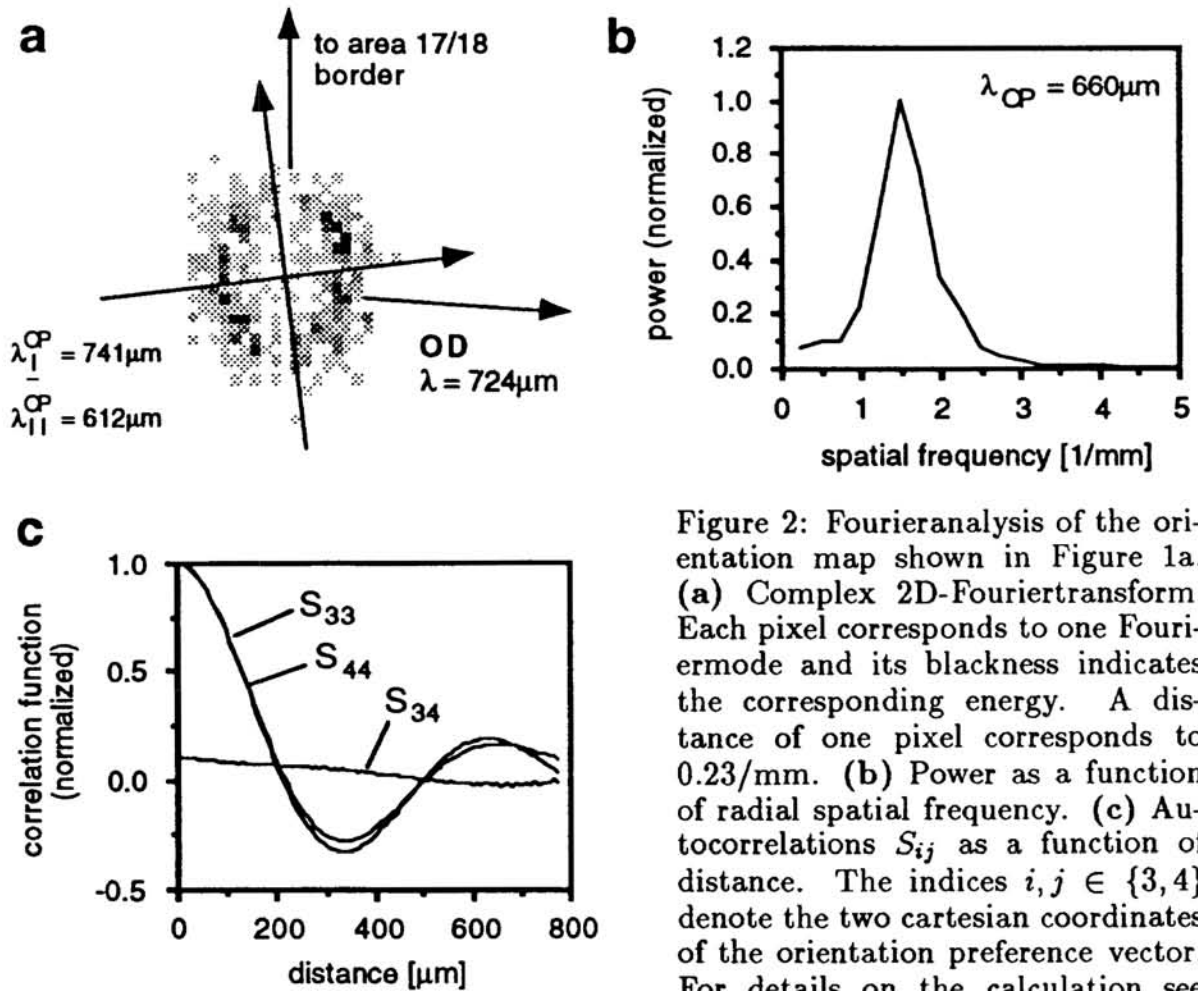

Figure 2: Fourieranalysis of the orientation map shown in Figure 1a. (a) Complex 2D-Fouriertransform. Each pixel corresponds to one Fouriermode and its blackness indicates the corresponding energy. A distance of one pixel corresponds to 0.23/mm. (b) Power as a function of radial spatial frequency. (c) Autocorrelations $S_{ij}$ as a function of distance. The indices $i, j \in \{3, 4\}$ denote the two cartesian coordinates of the orientation preference vector. For details on the calculation see [13, 15].

singularities, and fractures (Fig. 1c). The ocular dominance map shows its typical pattern of alternating bands.

Figure 2a shows the result of a complex 2D Fourier transform of the orientation map shown in Figure 1a. Like for maps recorded from adult monkeys [13] the spectrum is characterized by a slightly elliptical band of modes which is centered at the origin. The major axis approximately aligns with the axis parallel to the border to area 18 as well as with the ocular dominance bands. Therefore, like in the adults, the orientation map is stretched perpendicular to the ocular dominance bands, apparently to adjust to the wider spacing.

When one neglects the slight anisotropy of the Fourier spectra one can estimate a power spectrum by averaging the squared Fourier amplitudes for similar frequencies. The result is a pronounced peak whose location is given by the characteristic frequency of the orientation map (Fig. 2b). As a consequence, autocorrelation functions have a Mexican-hat shape (Fig. 2c), much like it has been reported for adults [13, 15].

In summary, the basic features of the patterns of orientation and ocular dominance are established as early as 3.5 weeks of age. Data which were recorded from four

Table 1: Characteristic wavelengths ($\lambda_{OD}$) and signal strengths ($\sigma_{OD}$) for the ocular dominance pattern, as well as characteristic wavelengths ($\lambda_{OP}$), density of $+180^o$-singularities ($\rho_+$), density of $-180^o$-singularities ($\rho_-$), total density of singularities ($\rho$), and percentage of area covered by linear zones ($a_{lin}$) for the orientation pattern as a function of age.

| age (weeks) | $\sigma_{OD}$ | $\lambda_{OD}$ ($\mu$m) | $\lambda_{OP}$ ($\mu$m) | $\rho_+$ (mm$^{-2}$) | $\rho_-$ (mm$^{-2}$) | $\rho$ (mm$^{-2}$) | $a_{lin}$ (% area) |
|---|---|---|---|---|---|---|---|
| 3.5 | 0.92 | 686 | 660 | 3.9 | 3.9 | 7.8 | 47 |
| 5.5 | 0.96 | 730 | 714 | 3.7 | 3.7 | 7.4 | 49 |
| 7.5 | 0.66 | 870 | 615 | 4.5 | 4.5 | 9.0 | 45 |
| 14 | 1.23 | 917 | 700 | 3.9 | 3.8 | 7.7 | 36 |
| adult | 1.36 | 950 | 768 | 3.9 | 3.8 | 7.7 | 43 |

other infants ranging from 5.5 to 14 weeks (not shown) confirm the above findings.

## 2.2   CHARACTERISTIC WAVELENGTHS AND SIGNAL STRENGTH

A more detailled analysis of the recorded patterns, however, reveals changes of certain features with age. Table 1 shows the changes in the typical wavelength of the orientation and ocular dominance patterns as well as the (normalized) ocular dominance signal strength with age. The strength of the ocular dominance signal increases by a factor of 1.5 between 3.5 weeks and adulthood, a fact, which could be explained by the still ongoing segregation of fibers within layer IVc.

The spacing of the ocular dominance columns increases by approximately 30% between 3.5 weeks and adulthood. This change in spacing would be consistent with the growth of cortical surface area during this period [16] if one assumes that cortex grows anisotropically in the direction perpendicular to the ocular dominance bands. Interestingly, the characteristic wavelengths of the orientation patterns do not exhibit such an increase. The wavelengths for the patterns recorded from the different infants are close to the "adult" values. More evidence for a stable orientation pattern is provided by the fact, that the density of singularities is approximately constant with age[2] and that the percentage of cortical area covered by linear zones does neither increase nor decrease. Hence we are left with the puzzle that at least the pattern of orientation does not follow cortical growth.

## 2.3   CORRELATIONS BETWEEN THE ORIENTATION AND OCULAR DOMINANCE MAPS

Figure 3 shows a contour plot representation of the pattern of orientation preference in overlay with the borders of the ocular dominance bands for the 3.5 week old animal. Iso-orientation contours (thin lines) indicate intervals of $15^o$. Thick lines indicate the border of the ocular dominance bands. From visual inspection it is

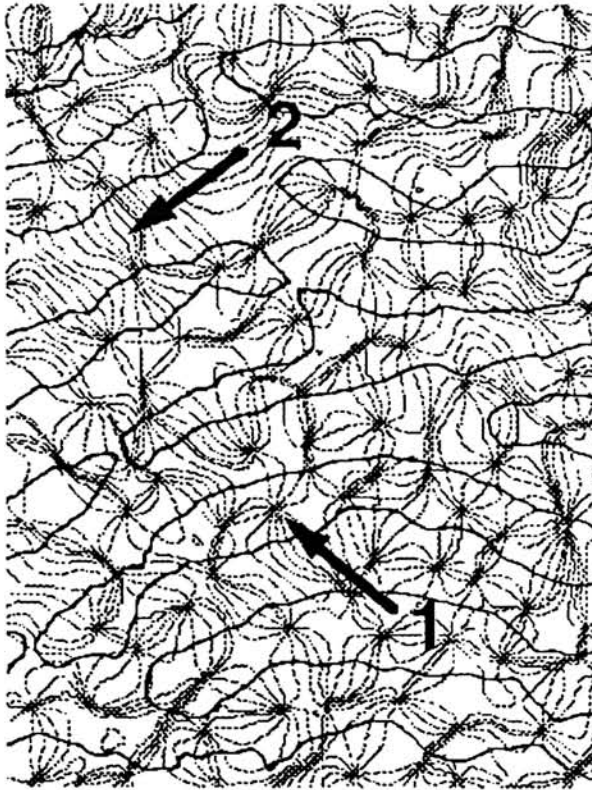

Figure 3: Contour plot representation of the orientation map shown in Figure 1a in overlay with the borders of the ocular dominance bands taken from Figure 1b. Iso-orientation lines (thin lines) indicate intervals of $15°$. The borders of the ocular dominance bands are indicated by thick lines.

already apparent that singularities have a strong tendency to align with the center of the ocular dominance bands (arrow 1) and that in the linear zones (arrow 2), where iso-orientation bands exist, these bands intersect ocular dominance bands at angles close to $90°$ most of the time.

Table 2 shows a quantitative analysis of the local intersection angle. Percentage of area covered by linear zones (cf. [12] for details of the calculation) is given for regions, where orientation bands intersect ocular dominance bands within $18°$ of perpendicular, and regions where they intersect within $18°$ of parallel. For all of the animals investigated the percentages are two to four times higher for regions, where orientation bands intersect ocular dominance bands at angles close to $90°$, much like it has been observed in adults [12]. In particular there is no consistent trend with age: the correlations between the orientation and ocular dominance maps are established as early as 3.5 weeks of age.

| age (weeks) | $a_{lin}^{perp}$ (%area) | $a_{lin}^{par}$ (% area) |
|---|---|---|
| 3.5 | 15.9 | 4.1 |
| 5.5 | 12.2 | 6.8 |
| 7.5 | 13.3 | 6.2 |
| 14 | 12.4 | 3.7 |
| adult | 18.0 | 2.7 |

Table 2: Percentage of area covered by linear zones as a function of age for regions, where orientation bands intersect ocular dominance bands within $18°$ of perpendicular ($a_{lin}^{perp}$), and regions where they intersect within $18°$ of parallel ($a_{lin}^{par}$) (cf. [12] for details of the calculation).

# 3   CONCLUSIONS AND RELATION TO MODELLING

In summary, our results provide evidence that the pattern of orientation is established at a time when the pattern of ocular dominance is still developing. However, they provide also evidence for the fact that the pattern of orientation is not linked to cortical growth. This latter finding still needs to be firmly established in studies where the development of orientation is followed in one and the same animal. But if it is taken seriously the consequence would be that orientation preferences may shift and that pairs of singularities are formed. The early presence of strong correlations between both maps indicate that the development of orientation and ocular dominance are not independent processes. Both patterns have to adjust to each other while cortex is growing. It, therefore, seems as if the third hypothesis is true (see Introduction) which states that both patterns develop independently and adjust to each other in the late stages of development. As has been shown in [13, 15] and is suggested in [7, 14] these processes are certainly in the realm of models based on Hebbian learning.

Many features of the orientation and ocular dominance maps are present at an age when the visual system of the monkey is still immature [8, 11]. In particular, they are present at a time when spatial vision is strongly impared. Consequently, it seems unlikely that the development of these features as well as of the correlations between both patterns requires high acuity form vision, and models which try to predict the structure of these maps from the structure of visual images [1, 2, 10] have to take this fact into account. The early development of orientation preference and its correlations with ocular dominance make it also seem more likely that their development may me an innate process and may not require extensive visual experience. Further experiments, however, are needed to settle these questions.

## Acknowledgements

This work was funded in part by the Klingenstein Foundation, the McKnight Foundation, the New England Primate Research Center (P51RRO168-31), the Seaver Institute, and the Howard Hughes Medical Institute. We thank Terry Sejnowski, Peter Dayan, and Rich Zemel for useful comments on the manuscript. Linda Ascomb, Jaqueline Mack, and Gina Quinn provided excellent technical assistance.

## Footnotes

[1] For all animals orientation and ocular dominance were recorded from a region close to the border to area 18 and close to midline.

[2]Note that both types of singularities appear in equal numbers.

# References

[1] H. G. Barrow and A. J. Bray. Activity induced color blob formation. In I. Alexander and J. Taylor, editors, *Artificial Neural Networks II*, pages 5–9. Elsevier Publishers, 1992.

[2] H. G. Barrow and A. J. Bray. A model of the adaptive development of complex cortical cells. In I. Alexander and J. Taylor, editors, *Artificial Neural Networks II*, pages 1–4. Elsevier Publishers, 1992.

[3] E. Bartfeld and A. Grinvald. Relationships between orientation-preference pinwheels, cytochrome oxidase blobs, and ocular-dominance columns in primate striate cortex. *Proc. Natl. Acad. Sci. USA*, 89:11905–11909, 1992.

[4] G. G. Blasdel. Differential imaging of ocular dominance and orientation selectivity in monkey striate cortex. *J. Neurosci.*, 12:3117-3158, 1992.

[5] G. G. Blasdel. Orientation selectivity, preference, and continuity in monkey striate cortex. *J. Neurosci.*, 12:3139-3161, 1992.

[6] G. G. Blasdel and G. Salama. Voltage sensitive dyes reveal a modular organization in monkey striate cortex. *Nature*, 321:579-585, 1986.

[7] R. Durbin and G. Mitchison. A dimension reduction framework for understanding cortical maps. *Nature*, 343:644-647, 1990.

[8] L. Kiorpes and T. Movshon. Behavioural analysis of visual development. In J. R. Coleman, editor, *Development of Sensory Systems in Mammals*, pages 125-154. John Wiley, 1990.

[9] S. LeVay, D. H. Hubel, and T. N. Wiesel. The development of ocular dominance columns in normal and visually deprived monkeys. *J. Comp. Neurol.*, 191:1-51, 1980.

[10] Y. Liu and H. Shouval. Principal component analysis of natural images – an analytic solution. Preprint.

[11] T. Movshon and L. Kiorpes. Biological limits on visual development in primates. In K. Simon, editor, *Handbook of Infant Vision*. Oxford University Press, 1993. in press.

[12] K. Obermayer and G. G. Blasdel. Geometry of orientation and ocular dominance columns in monkey striate cortex. *J. Neurosci.*, 13:4114-4129, 1993.

[13] K. Obermayer, G. G. Blasdel, and K. Schulten. A statistical mechanical analysis of self-organization and pattern formation during the development of visual maps. *Phys. Rev. A15*, 45:7568-7589, 1992.

[14] K. Obermayer, H. Ritter, and K. Schulten. A principle for the formation of the spatial structure of cortical feature maps. *Proc. Natl. Acad. Sci. USA*, 87:8345-8349, 1990.

[15] K. Obermayer, K. Schulten, and G. G. Blasdel. A comparison of a neural network model for the formation of brain maps with experimental data. In D. S. Touretzky and R. Lippman, editors, *Advances in Neural Information Processing Systems 4*, pages 83-90. Morgan Kaufmann Publishers, 1992.

[16] D. Purves and A. LaMantia. Development of blobs in the visual cortex of macaques. *J. Comp. Neurol.*, 332:1-7, 1993.

[17] N. Swindale. A model for the coordinated development of columnar systems in primate striate cortex. *Biol. Cybern.*, 66:217-230, 1992.

[18] D. Y. Tso, R. D. Frostig, E. E. Lieke, and A. Grinvald. Functional organization of primate visual cortex revealed by high resolution optical imaging. *Science*, 249:417-420, 1990.

[19] T. N. Wiesel and D. H. Hubel. Ordered arrangement of orientation columns in monkeys lacking visual experience. *J. Comp. Neurol.*, 158:307-318, 1974.